# Experts in a Markov Decision Process

**Eyal Even-Dar**
Computer Science
Tel-Aviv University
evend@post.tau.ac.il

**Sham M. Kakade**
Computer and Information Science
University of Pennsylvania
skakade@linc.cis.upenn.edu

**Yishay Mansour** [*]
Computer Science
Tel-Aviv University
mansour@post.tau.ac.il

## Abstract

We consider an MDP setting in which the reward function is allowed to change during each time step of play (possibly in an adversarial manner), yet the dynamics remain fixed. Similar to the experts setting, we address the question of how well can an agent do when compared to the reward achieved under the best stationary policy over time. We provide *efficient* algorithms, which have regret bounds with *no dependence* on the size of state space. Instead, these bounds depend only on a certain horizon time of the process and logarithmically on the number of actions. We also show that in the case that the dynamics change over time, the problem becomes computationally hard.

## 1   Introduction

There is an inherent tension between the objectives in an expert setting and those in a reinforcement learning setting. In the experts problem, during every round a learner chooses one of $n$ decision making experts and incurs the loss of the chosen expert. The setting is typically an adversarial one, where Nature provides the examples to a learner. The standard objective here is a myopic, backwards looking one — in retrospect, we desire that our performance is not much worse than had we chosen any *single* expert on the sequence of examples provided by Nature. In contrast, a reinforcement learning setting typically makes the much stronger assumption of a fixed environment, typically a Markov decision process (MDP), and the forward looking objective is to maximize some measure of the future reward with respect to this fixed environment.

The motivation of this work is to understand how to *efficiently* incorporate the benefits of existing experts algorithms into a more adversarial reinforcement learning setting, where certain aspects of the environment could change over time. A naive way to implement an experts algorithm is to simply associate an expert with each fixed policy. The running time of such algorithms is polynomial in the number of experts and the regret (the difference from the optimal reward) is logarithmic in the number of experts. For our setting the number of policies is huge, namely $\#\text{actions}^{\#\text{states}}$, which renders the naive experts approach computationally infeasible.

Furthermore, straightforward applications of standard regret algorithms produce regret bounds which are logarithmic in the number of policies, so they have linear dependence

---
[*]This work was supported in part by the IST Programme of the European Community, under the PASCAL Network of Excellence, IST-2002-506778, by a grant from the Israel Science Foundation and an IBM faculty award. This publication only reflects the authors' views.

on the number of states. We might hope for a more effective regret bound which has *no dependence* on the size of state space (which is typically large).

The setting we consider is one in which the dynamics of the environment are known to the learner, but the reward function can change over time. We assume that after each time step the learner has complete knowledge of the previous reward functions (over the entire environment), but does not know the future reward functions.

As a motivating example one can consider taking a long road-trip over some period of time $T$. The dynamics, namely the roads, are fixed, but the road conditions may change frequently. By listening to the radio, one can get (effectively) instant updates of the road and traffic conditions. Here, the task is to minimize the cost during the period of time $T$. Note that at each time step we select one road segment, suffer a certain delay, and need to plan ahead with respect to our current position.

This example is similar to an adversarial shortest path problem considered in Kalai and Vempala [2003]. In fact Kalai and Vempala [2003], address the computational difficulty of handling a large number of experts under certain linear assumptions on the reward functions. However, their algorithm is not directly applicable to our setting, due to the fact that in our setting, decisions must be made with respect to the *current* state of the agent (and the reward could be changing frequently), while in their setting the decisions are only made with respect to a single state.

McMahan et al. [2003] also considered a similar setting — they also assume that the reward function is chosen by an adversary and that the dynamics are fixed. However, they assume that the cost functions come from a finite set (but are not observable) and the goal is to find a min-max solution for the related stochastic game.

In this work, we provide *efficient* ways to incorporate existing best experts algorithms into the MDP setting. Furthermore, our loss bounds (compared to the best constant policy) have *no dependence* on the number of states and depend only on on a certain horizon time of the environment and $\log(\#actions)$. There are two sensible extensions of our setting. The first is where we allow Nature to change the dynamics of the environment over time. Here, we show that it becomes NP-Hard to develop a low regret algorithm even for oblivious adversary. The second extension is to consider one in which the agent only observes the rewards for the states it actually visits (a generalization of the multi-arm bandits problem). We leave this interesting direction for future work.

## 2   The Setting

We consider an MDP with state space $S$; initial state distribution $d_1$ over $S$; action space $A$; state transition probabilities $\{P_{sa}(\cdot)\}$ (here, $P_{sa}$ is the next-state distribution on taking action $a$ in state $s$); and a sequence of reward functions $r_1, r_2, \ldots r_T$, where $r_t$ is the (bounded) reward function at time step $t$ mapping $S \times A$ into $[0, 1]$.

The goal is to maximize the sum of undiscounted rewards over a $T$ step horizon. We assume the agent has complete knowledge of the transition model $P$, but at time $t$, the agent only knows the past reward functions $r_1, r_2, \ldots r_{t-1}$. Hence, an algorithm $\mathcal{A}$ is a mapping from $S$ and the previous reward functions $r_1, \ldots r_{t-1}$ to a probability distribution over actions, so $\mathcal{A}(a|s, r_1, \ldots r_{t-1})$ is the probability of taking action $a$ at time $t$.

We define the return of an algorithm $\mathcal{A}$ as:

$$V_{r_1, r_2, \ldots r_T}(\mathcal{A}) = \frac{1}{T} \mathrm{E} \left[ \sum_{t=1}^{T} r_t(s_t, a_t) \Big| d_1, \mathcal{A} \right]$$

where $a_t \sim \mathcal{A}(a|s_t, r_1, \ldots r_{t-1})$ and $s_t$ is the random variable which represents the state

at time $t$, starting from initial state $s_1 \sim d_1$ and following actions $a_1, a_2, \ldots a_{t-1}$. Note that we keep track of the expectation and not of a specific trajectory (and our algorithm specifies a distribution over actions at *every* state and at *every* time step $t$).

Ideally, we would like to find an $\mathcal{A}$ which achieves a large reward $V_{r_1, \ldots r_T}(\mathcal{A})$ *regardless* of how the adversary chooses the reward functions. In general, this of course is not possible, and, as in the standard experts setting, we desire that our algorithm competes favorably against the best fixed stationary policy $\pi(a|s)$ in hindsight.

# 3 An MDP Experts Algorithm

## 3.1 Preliminaries

Before we provide our algorithm a few definitions are in order. For every stationary policy $\pi(a|s)$, we define $P^\pi$ to be the transition matrix induced by $\pi$, where the component $[P^\pi]_{s,s'}$ is the transition probability from $s$ to $s'$ under $\pi$. Also, define $d_{\pi,t}$ to be the state distribution at time $t$ when following $\pi$, *ie*

$$d_{\pi,t} = d_1 (P^\pi)^t$$

where we are treating $d_1$ as a row vector here.

**Assumption 1** *(Mixing) We assume the transition model over states, as determined by $\pi$, has a well defined stationary distribution, which we call $d_\pi$. More formally, for every initial state $s$, $d_{\pi,t}$ converges to $d_\pi$ as $t$ tends to infinity and $d_\pi P^\pi = d_\pi$. Furthermore, this implies there exists some $\tau$ such that for* all *policies $\pi$, and distributions $d$ and $d'$,*

$$\|dP^\pi - d'P^\pi\|_1 \le e^{-1/\tau} \|d - d'\|_1$$

*where $\|x\|_1$ denotes the $l_1$ norm of a vector $x$. We refer to $\tau$ as the* mixing time *and assume that $\tau > 1$.*

The parameter $\tau$ provides a bound on the planning horizon timescale, since it implies that *every* policy achieves close to its average reward in $O(\tau)$ steps [1]. This parameter also governs how long it effectively takes to switch from one policy to another (after time $O(\tau)$ steps there is little information in the state distribution about the previous policy).

This assumption allows us to define the average reward of policy $\pi$ in an MDP with reward function $r$ as:

$$\eta_r(\pi) = \mathrm{E}_{s \sim d_\pi, a \sim \pi(a|s)}[r(s,a)]$$

and the value, $Q_{\pi,r}(s,a)$, is defined as

$$Q_{\pi,r}(s,a) \equiv \mathrm{E}\left[\sum_{t=1}^\infty (r(s_t,a_t) - \eta_r(\pi)) \,\Big|\, s_1 = s, a_1 = a, \pi\right]$$

where and $s_t$ and $a_t$ are the state and actions at time $t$, after starting from state $s_1 = s$ then deviating with an immediate action of $a_1 = a$ and following $\pi$ onwards. We slightly abuse notation by writing $Q_{\pi,r}(s,\pi') = \mathrm{E}_{a \sim \pi'(a|s)}[Q_{\pi,r}(s,a)]$. These values satisfy the well known recurrence equation:

$$Q_{\pi,r}(s,a) = r(s,a) - \eta_r(\pi) + \mathrm{E}_{s' \sim P_{sa}}[Q_\pi(s',\pi)] \tag{1}$$

where $Q_\pi(s',\pi)$ is the next state value (without deviation).

If $\pi^*$ is an optimal policy (with respect to $\eta_r$), then, as usual, we define $Q_r^*(s, a)$ to be the value of the optimal policy, *ie* $Q_r^*(s, a) = Q_{\pi^*, r}(s, a)$.

We now provide two useful lemmas. It is straightforward to see that the previous assumption implies a rate of convergence to the stationary distribution that is $O(\tau)$, for all policies. The following lemma states this more precisely.

**Lemma 2** *For all policies $\pi$,*

$$\|d_{\pi,t} - d_\pi\|_1 \le 2e^{-t/\tau} .$$

*Proof.* Since $\pi$ is stationary, we have $d_\pi P^\pi = d_\pi$, and so

$$\|d_{\pi,t} - d_\pi\|_1 = \|d_{\pi,t-1}P^\pi - d_\pi P^\pi\|_1 \le \|d_{\pi,t-1} - d_\pi\|_1 e^{-1/\tau}$$

which implies $\|d_{\pi,t} - d_\pi\|_1 \le \|d_1 - d_\pi\|_1 e^{-t/\tau}$. The claim now follows since, for all distributions $d$ and $d'$, $\|d - d'\|_1 \le 2$. $\quad\square$

The following derives a bound on the $Q$ values as a function of the mixing time.

**Lemma 3** *For all reward functions $r$, $Q_{\pi,r}(s, a) \le 3\tau$ .*

*Proof.* First, let us bound $Q_{\pi,r}(s, \pi)$, where $\pi$ is used on the first step. For all $t$, including $t = 1$, let $d_{\pi,s,t}$ be the state distribution at time $t$ starting from state $s$ and following $\pi$. Hence, we have

$$
\begin{aligned}
Q_{\pi,r}(s, \pi) &= \sum_{t=1}^{\infty} \left( E_{s' \sim d_{\pi,s,t}, a \sim \pi}[r(s', a)] - \eta_r(\pi)) \right) \\
&\le \sum_{t=1}^{\infty} \left( E_{s' \sim d_\pi, a \sim \pi}[r(s', a)] - \eta_r(\pi) + 2e^{-t/\tau} \right) \\
&= \sum_{t=1}^{\infty} 2e^{-t/\tau} \le \int_0^\infty 2e^{-t/\tau} = 2\tau
\end{aligned}
$$

Using the recurrence relation for the values, we know $Q_{\pi,r}(s, a)$ could be at most $1$ more than the above. The result follows since $1 + 2\tau \le 3\tau$ $\quad\square$

### 3.2 The Algorithm

Now we provide our main result showing how to use any generic experts algorithm in our setting. We associate each state with an experts algorithm, and the expert for each state is responsible for choosing the actions at that state. The immediate question is what loss function should we feed to each expert. It turns out $Q_{\pi_t, r_t}$ is appropriate. We now assume that our experts algorithm achieves a performance comparable to the best constant action.

**Assumption 4** *(Black Box Experts) We assume access to an optimized best expert algorithm which guarantees that for any sequence of loss functions $c_1, c_2, \ldots c_T$ over actions $A$, the algorithm selects a distribution $q_t$ over $A$ (using only the previous loss functions $c_1, c_2, \ldots c_{t-1}$) such that*

$$\sum_{t=1}^{T} E_{a \sim q_t}[c_t(a)] \le \sum_{t=1}^{T} c_t(a) + M\sqrt{T \log |A|},$$

*where $\|c_t(a)\| \le M$. Furthermore, we also assume that decision distributions do not change quickly:*

$$\|q_t - q_{t+1}\|_1 \le \sqrt{\frac{\log |A|}{t}}$$

These assumptions are satisfied by the multiplicative weights algorithms. For instance, the algorithm in Freund and Schapire [1999] is such that the for each decision $a$, $|\log q_t(a) - \log q_{t+1}(a)|$ changes by $O(\sqrt{\frac{\log |A|}{t}})$, which implies the weaker $l_1$ condition above.

In our setting, we have an experts algorithm associated with *every* state $s$, which is fed the loss function $Q_{\pi_t, r_t}(s, \cdot)$ at time $t$. The above assumption then guarantees that at every state $s$ for every action $a$ we have that

$$\sum_{t=1}^{T} Q_{\pi_t, r_t}(s, \pi_t) \leq \sum_{t=1}^{T} Q_{\pi_t, r_t}(s, a) + 3\tau \sqrt{T \log |A|}$$

since the loss function $Q_{\pi_t, r_t}$ is bounded by $3\tau$, and that

$$|\pi_t(\cdot|s) - \pi_{t+1}(\cdot|s)|_1 \leq \sqrt{\frac{\log |A|}{t}}$$

As we shall see, it is important that this 'slow change' condition be satisfied. Intuitively, our experts algorithms will be using a similar policy for significantly long periods of time.

Also note that since the experts algorithms are associated with each state and each of the $N$ experts chooses decisions out of $A$ actions, the algorithm is efficient (polynomial in $N$ and $A$, assuming that that the black box uses a reasonable experts algorithm).

We now state our main theorem.

**Theorem 5** *Let $\mathcal{A}$ be the MDP experts algorithm. Then for all reward functions $r_1, r_2, \ldots r_T$ and for all stationary policies $\pi$,*

$$V_{r_1, r_2, \ldots r_T}(\mathcal{A}) \geq V_{r_1, r_2, \ldots r_T}(\pi) - 8\tau^2 \sqrt{\frac{\log |A|}{T}} - 3\tau \sqrt{\frac{\log |A|}{T}} - \frac{4\tau}{T}$$

As expected, the regret goes to $0$ at the rate $O(1/\sqrt{T})$, as is the case with experts algorithms. Importantly, note that the bound does *not depend* on the size of the state space.

### 3.3 The Analysis

The analysis is naturally divided into two parts. First, we analyze the performance of the algorithm in an idealized setting, where the algorithm instantaneously obtains the average reward of its current policy at each step. Then we take into account the slow change of the policies to show that the actual performance is similar to the instantaneous performance.

**An Idealized Setting:** Let us examine the case in which at each time $t$, when the algorithm uses $\pi_t$, it immediately obtains reward $\eta_{r_t}(\pi_t)$. The following theorem compares the performance of our algorithms to that of a fixed constant policy in this setting.

**Theorem 6** *For all sequences $r_1, r_2, \ldots r_T$, the MDP experts algorithm have the following performance bound. For all $\pi$,*

$$\sum_{t=1}^{T} \eta_{r_t}(\pi_t) \geq \sum_{t=1}^{T} \eta_{r_t}(\pi) - 3\tau \sqrt{T \log |A|}$$

*where $\pi_1, \pi_2, \ldots \pi_T$ is the sequence of policies generated by $\mathcal{A}$ in response to $r_1, r_2, \ldots r_T$.*

Next we provide a technical lemma, which is a variant of a result in Kakade [2003]

**Lemma 7** *For all policies $\pi$ and $\pi'$,*

$$\eta_r(\pi') - \eta_r(\pi) = \mathrm{E}_{s \sim d_{\pi'}}[Q_{\pi, r}(s, \pi') - Q_{\pi, r}(s, \pi)]$$

*Proof.* Note that by definition of stationarity, if the state distribution is at $d_{\pi'}$, then the next state distribution is also $d_{\pi'}$ if $\pi'$ is followed. More formally, if $s \sim d_{\pi'}$, $a \sim \pi'(a|s)$, and $s' \sim P_{sa}$, then $s' \sim d_{\pi'}$. Using this and equation 1, we have:

$$
\begin{aligned}
\mathrm{E}_{s \sim d_{\pi'}}[Q_{\pi,r}(s,\pi')] &= \mathrm{E}_{s \sim d_{\pi'}, a \sim \pi'}[Q_{\pi,r}(s,a)] \\
&= \mathrm{E}_{s \sim d_{\pi'}, a \sim \pi'}[r(s,a) - \eta_r(\pi) + \mathrm{E}_{s' \sim P_{sa}}[Q_\pi(s',\pi)] \\
&= \mathrm{E}_{s \sim d_{\pi'}, a \sim \pi'}[r(s,a) - \eta_r(\pi)] + \mathrm{E}_{s \sim d_{\pi'}}[Q_\pi(s,\pi)] \\
&= \eta_r(\pi') - \eta_r(\pi) + \mathrm{E}_{s \sim d_{\pi'}}[Q_\pi(s,\pi)]
\end{aligned}
$$

Rearranging terms leads to the result. $\qquad\square$

The lemma shows why our choice to feed each experts algorithm $Q_{\pi_t,r_t}$ was appropriate. Now we complete the proof of the above theorem.

*Proof.* Using the assumed regret in assumption 4,

$$
\begin{aligned}
\sum_{t=1}^{T} \eta_{r_t}(\pi) - \sum_{t=1}^{T} \eta_{r_t}(\pi_t) &= \sum_{t=1}^{T} \mathrm{E}_{s \sim d_\pi}[Q_{\pi_t,r_t}(s,\pi) - Q_{\pi_t,r_t}(s,\pi_t)] \\
&= \mathrm{E}_{s \sim d_\pi}[\sum_{t=1}^{T} Q_{\pi_t,r_t}(s,\pi) - Q_{\pi_t,r_t}(s,\pi_t)] \\
&\leq \mathrm{E}_{s \sim d_\pi}[3\tau\sqrt{T\log A}] \\
&= 3\tau\sqrt{T\log A}
\end{aligned}
$$

where we used the fact that $d_\pi$ does not depend on the time in the second step. $\qquad\square$

**Taking Mixing Into Account:** This subsection relates the values $V$ to the sums of average reward used in the idealized setting.

**Theorem 8** *For all sequences $r_1, r_2, \ldots r_T$ and for all $\mathcal{A}$*

$$
|V_{r_1,r_2,\ldots r_T}(\mathcal{A}) - \frac{1}{T}\sum_{t=1}^{T} \eta_{r_t}(\pi_t)| \leq 4\tau^2\sqrt{\frac{\log|A|}{T}} + \frac{2\tau}{T}
$$

*where $\pi_1, \pi_2, \ldots \pi_T$ is the sequence of policies generated by $\mathcal{A}$ in response to $r_1, r_2, \ldots r_T$.*

Since the above holds for all $\mathcal{A}$ (including those $\mathcal{A}$ which are the constant policy $\pi$), then combining this with Theorem 6 (once with $\mathcal{A}$ and once with $\pi$) completes the proof of Theorem 5. We now prove the above.

The following simple lemma is useful and we omit the proof. It shows how close are the next state distributions when following $\pi_t$ rather than $\pi_{t+1}$.

**Lemma 9** *Let $\pi$ and $\pi'$ be such that $\|\pi(\cdot|s) - \pi'(\cdot|s)\|_1 \leq \epsilon$. Then for any state distribution $d$, we have $\|dP^\pi - dP^{\pi'}\|_1 \leq \varepsilon$.*

Analogous to the definition of $d_{\pi,t}$, we define $d_{\mathcal{A},t}$

$$
d_{\mathcal{A},t} = \Pr[s_t = s|d_1, \mathcal{A}]
$$

which is the probability that the state at time $t$ is $s$ given that $\mathcal{A}$ has been followed.

**Lemma 10** *Let $\pi_1, \pi_2, \ldots \pi_T$ be the sequence of policies generated by $\mathcal{A}$ in response to $r_1, r_2, \ldots r_T$. We have*

$$
\|d_{\mathcal{A},t} - d_{\pi_t}\|_1 \leq 2\tau^2\sqrt{\frac{\log|A|}{t}} + 2e^{-t/\tau}
$$

*Proof.* Let $k \leq t$. Using our experts assumption, it is straightforward to see that that the change in the policy over $k$ steps is $|\pi_k(\cdot|s) - \pi_t(\cdot|s)|_1 \leq (t-k)\sqrt{\log|A|/t}$. Using this with $d_{\mathcal{A},k} = d_{\mathcal{A},k-1}P(\pi_k)$ and $d_{\pi_t}P^{\pi_t} = d_{\pi_t}$, we have

$$
\begin{aligned}
\|d_{\mathcal{A},k} - d_{\pi_t}\|_1 &= \|d_{\mathcal{A},k-1}P^{\pi_k} - d_{\pi_t}\|_1 \\
&\leq \|d_{\mathcal{A},k-1}P^{\pi_t} - d_{\pi_t}\|_1 + \|d_{\mathcal{A},k-1}P^{\pi_k} - d_{\mathcal{A},k-1}P^{\pi_t}\|_1 \\
&\leq \|d_{\mathcal{A},k-1}P^{\pi_t} - d_{\pi_t}P^{\pi_t}\|_1 + 2(t-k)\sqrt{\log|A|/t} \\
&\leq e^{-1/\tau}\|d_{\mathcal{A},k-1} - d_{\pi_t}\|_1 + 2(t-k)\sqrt{\log|A|/t}
\end{aligned}
$$

where we have used the last lemma in the third step and our contraction assumption 1 in the second to last step. Recursing on the above equation leads to:

$$
\begin{aligned}
\|d_{\mathcal{A},t} - d_{\pi_t}\| &\leq 2\sqrt{\log|A|/t}\sum_{k=t}^{2}(t-k)e^{-(t-k)/\tau} + e^{-t/\tau}\|d_1 - d_{\pi_t}\| \\
&\leq 2\sqrt{\log|A|/t}\sum_{k=1}^{\infty}ke^{-k/\tau} + 2e^{-t/\tau}
\end{aligned}
$$

The sum is bounded by an integral from 0 to $\infty$, which evaluates to $\tau^2$. $\quad\square$

We are now ready to complete the proof of Theorem 8.

*Proof.* By definition of $V$,

$$
\begin{aligned}
V_{r_1,r_2,\ldots r_T}(\mathcal{A}) &= \frac{1}{T}\sum_{t=1}^{T}E_{s\sim d_{\mathcal{A},t},a\sim\pi_t}[r_t(s,a)] \\
&\leq \frac{1}{T}\sum_{t=1}^{T}E_{s\sim d_{\pi_t},a\sim\pi_t}[r_t(s,a)] + \frac{1}{T}\sum_{t=1}^{T}\|d_{\mathcal{A},t} - d_{\pi_t}\|_1 \\
&\leq \frac{1}{T}\sum_{t=1}^{T}\eta_{r_t}(\pi_t) + \frac{1}{T}\sum_{t=1}^{T}\left(2\tau^2\sqrt{\frac{\log|A|}{t}} + 2e^{-t/\tau}\right) \\
&\leq \frac{1}{T}\sum_{t=1}^{T}\eta_{r_t}(\pi_t) + 4\tau^2\sqrt{\frac{\log|A|}{T}} + \frac{2\tau}{T}
\end{aligned}
$$

where we have bounded the sums by integration in the second to last step. A symmetric argument leads to the result. $\quad\square$

## 4   A More Adversarial Setting

In this section we explore a different setting, the *changing dynamics model*. Here, in each timestep $t$, an oblivious adversary is allowed to choose both the reward function $r_t$ and the transition model $P_t$ — the model that determines the transitions to be used at timestep $t$. After each timestep, the agent receives complete knowledge of both $r_t$ and $P_t$. Furthermore, we assume that $P_t$ is deterministic, so we do not concern ourselves with mixing issues. In this setting, we have the following hardness result. We let $R_t^*(M)$ be the optimal average reward obtained by a stationary policy for times $[1, t]$.

**Theorem 11** *In the changing dynamics model, if there exists a polynomial time online algorithm (polynomial in the problem parameters) such that, for any MDP, has an expected average reward larger than $(0.875 + \varepsilon)R_t^*(M)$, for some $\varepsilon > 0$ and $t$, then $P = NP$.*

The following lemma is useful in the proof and uses the fact that it is hard to approximate MAX3SAT within any factor better than $0.875$ (Hastad [2001]).

**Lemma 12** *Computing a stationary policy in the changing dynamics model with average reward larger than $(0.875 + \varepsilon)R^*(M)$, for some $\varepsilon > 0$, is NP-Hard.*

**Proof:** We prove it by reduction from 3-SAT. Suppose that the 3-SAT formula, $\phi$ has $m$ clauses, $C1, \ldots, C_m$, and $n$ literals, $x_1, \ldots, x_n$ then we reduce it to MDP with $n + 1$ states,$s_1, \ldots s_n, s_{n+1}$, two actions in each state, $0, 1$ and fixed dynamic for $3m$ steps which will be described later. We prove that a policy with average reward $p/3$ translates to an assignment that satisfies $p$ fraction of $\phi$ and vice versa. Next we describe the dynamics. Suppose that $C_1$ is $(x_1 \vee \neg x_2 \vee x_7)$ and $C_2$ is $(x_4 \vee \neg x_1 \vee x_7)$. The initial state is $s_1$ and the reward for action 0 is 0 and the agent moves to state $s_2$, for action 1 the reward is 1 and it moves to state $s_{n+1}$. In the second timestep the reward in $s_{n+1}$ is 0 for every action and the agents stay in it; in state $s_2$ if the agent performs action 0 then it obtains reward 1 and move to state $s_{n+1}$ otherwise it obtains reward 0 and moves to state $s_7$. In the next timestep the reward in $s_{n+1}$ is 0 for every action and the agents moves to $x_4$, the reward in $s_7$ is 1 for action 1 and zero for action 0 and moves to $s_4$ for both actions. The rest of the construction is done identically. Note that time interval $[3(\ell - 1) + 1, 3\ell]$ corresponds to $C_\ell$ and that the reward obtained in this interval is at most 1. We note that $\phi$ has an assignment $y_1, \ldots, y_n$ where $y_i = \{0, 1\}$ that satisfies $p$ fraction of it, if and only if $\pi$ which takes action $y_i$ in $s_i$ has average reward $p/3$. We prove it by looking on each interval separately and noting that if a reward 1 is obtained then there is an action $a$ that we take in one of the states which has reward 1 but this action corresponds to a satisfying assignment for this clause. $\square$

We are now ready to prove Theorem 11.

**Proof:** In this proof we make few changes from the construction given in Lemma 12. We allow the same clause to repeat few times, and its dynamics are described in $n$ steps and not in 3 steps, where in the $k$ step we move from $s_k$ to $s_{k+1}$ and obtains 0 reward, unless the action "satisfies" the chosen clause, if it satisfies then we obtain an immediate reward 1, move to $s_{n+1}$ and stay there for $n - k - 1$ steps. After $n$ steps the adversary chooses uniformly at random the next clause. In the analysis we define the $n$ steps related to a clause as an iteration. The strategy defined by the algorithm at the $k$ iteration is the probability assigned to action $0/1$ at state $s_\ell$ just before arriving to $s_\ell$. Note that the strategy at each iteration is actually a stationary policy for $M$. Thus the strategy in each iteration defines an assignment for the formula. We also note that before an iteration the expected reward of the optimal stationary policy in the iteration is $k/(nm)$, where $k$ is the maximal number of satisfiable clauses and there are $m$ clauses, and we have $E[R^*(M)] = k/(nm)$. If we choose at random an iteration, then the strategy defined in that iteration has an expected reward which is larger than $(0.875 + \varepsilon)R^*(M)$, which implies that we can satisfy more than 0.875 fraction of satisfiable clauses, but this is impossible unless $P = NP$. $\square$

## Footnotes

[1] If this timescale is unreasonably large for some specific MDP, then one could artificially impose some horizon time and attempt to compete with those policies which mix in this horizon time, as done Kearns and Singh [1998].

# References

Y. Freund and R. Schapire. Adaptive game playing using multiplicative weights. *Games and Economic Behavior*, 29:79–103, 1999.

J. Hastad. Some optimal inapproximability results. *J. ACM*, 48(4):798–859, 2001.

S. Kakade. *On the Sample Complexity of Reinforcement Learning*. PhD thesis, University College London, 2003.

A. Kalai and S. Vempala. Efficient algorithms for on-line optimization. *Proceedings of COLT*, 2003.

M. Kearns and S. Singh. Near-optimal reinforcement learning in polynomial time. *Proceedings of ICML*, 1998.

H. McMahan, G. Gordon, and A. Blum. Planning in the presence of cost functions controlled by an adversary. In *In the 20th ICML*, 2003.
